# A Game-Theoretic Approach to Apprenticeship Learning

**Umar Syed**
Computer Science Department
Princeton University
35 Olden St
Princeton, NJ 08540-5233
usyed@cs.princeton.edu

**Robert E. Schapire**
Computer Science Department
Princeton University
35 Olden St
Princeton, NJ 08540-5233
schapire@cs.princeton.edu

## Abstract

We study the problem of an apprentice learning to behave in an environment with an unknown reward function by observing the behavior of an expert. We follow on the work of Abbeel and Ng [1] who considered a framework in which the true reward function is assumed to be a linear combination of a set of known and observable features. We give a new algorithm that, like theirs, is guaranteed to learn a policy that is nearly as good as the expert's, given enough examples. However, unlike their algorithm, we show that ours may produce a policy that is substantially better than the expert's. Moreover, our algorithm is computationally faster, is easier to implement, and can be applied even in the absence of an expert. The method is based on a game-theoretic view of the problem, which leads naturally to a direct application of the multiplicative-weights algorithm of Freund and Schapire [2] for playing repeated matrix games. In addition to our formal presentation and analysis of the new algorithm, we sketch how the method can be applied when the transition function itself is unknown, and we provide an experimental demonstration of the algorithm on a toy video-game environment.

## 1 Introduction

When an agent is faced with the task of learning how to behave in a stochastic environment, a common approach is to model the situation using a Markov Decision Process. An MDP consists of states, actions, rewards and a transition function. Once an MDP has been provided, the usual objective is to find a policy (i.e. a mapping from states to actions) that maximizes expected cumulative reward collected by the agent.

Building the MDP model is usually the most difficult part of this process. One reason is that it is often hard to correctly describe the environment's true reward function, and yet the behavior of the agent is quite sensitive to this description. In practice, reward functions are frequently tweaked and tuned to elicit what is thought to be the desired behavior. Instead of maximizing reward, another approach often taken is to observe and follow the behavior of an expert in the same environment. Learning how to behave by observing an expert has been called *apprenticeship learning*, with the agent in the role of the apprentice.

Abbeel and Ng [1] proposed a novel and appealing framework for apprenticeship learning. In this framework, the reward function, while unknown to the apprentice, is assumed to be equal to a linear combination of a set of known features. They argued that while it may be difficult to correctly describe the reward function, it is usually much easier to specify the features on which the reward function depends.

With this setting in mind, Abbeel and Ng [1] described an efficient algorithm that, given enough examples of the expert's behavior, produces a policy that does at least as well as the expert with respect to the unknown reward function. The number of examples their algorithm requires from the expert depends only moderately on the number of features.

While impressive, a drawback of their results is that the performance of the apprentice is both upper-*and* lower-bounded by the performance of the expert. Essentially, their algorithm is an efficient method for mimicking the expert's behavior. If the behavior of the expert is far from optimal, the same will hold for the apprentice.

In this paper, we take a somewhat different approach to apprenticeship learning that addresses this issue, while also significantly improving on other aspects of Abbeel and Ng's [1] results. We pose the problem as learning to play a two-player zero-sum game in which the apprentice chooses a policy, and the environment chooses a reward function. The goal of the apprentice is to maximize performance relative to the expert, even though the reward function may be adversarially selected by the environment with respect to this goal. A key property of our algorithm is that it is able to leverage prior beliefs about the relationship between the features and the reward function. Specifically, if it is known whether a feature is "good" (related to reward) or "bad" (inversely related to reward), then the apprentice can use that knowledge to improve its performance. As a result, our algorithm produces policies that can be significantly better than the expert's policy with respect to the unknown reward function, while at the same time are guaranteed to be no worse.

Our approach is based on a multiplicative weights algorithm for solving two-player zero-sum games due to Freund and Schapire [2]. Their algorithm is especially well-suited to solving zero-sum games in which the "game matrix" is extremely large. It turns out that our apprenticeship learning setting can be viewed as a game with this property.

Our results represent a strict improvement over those of Abbeel and Ng [1] in that our algorithm is considerably simpler, provides the same lower bound on the apprentice's performance relative to the expert, and removes the upper bound on the apprentice's performance. Moreover, our algorithm requires less computational expense – specifically, we are able to achieve their performance guarantee after only $O(\ln k)$ iterations, instead of the $O(k \ln k)$, where $k$ is the number of features on which the reward function depends. Additionally, our algorithm can be applied to a setting in which no examples are available from the expert. In that case, our algorithm produces a policy that is optimal in a certain conservative sense. We are also able to extend our algorithm to a situation where the MDP's transition function $\theta$ is unknown. We conducted experiments from a small car driving simulation that illustrate some of our theoretical findings.

Ratliff *et al* [3] formulated a related problem to apprenticeship learning, in which the goal is to find a reward function whose optimal policy is similar to the expert's policy. Quite different from our work, mimicking the expert was an explicit goal of their approach.

## 2   Preliminaries

Our problem setup largely parallels that outlined in Abbeel and Ng [1]. We are given an infinite-horizon Markov Decision Process in which the reward function has been replaced by a set of features. Specifically, we are given an MDP\R $M = (\mathcal{S}, \mathcal{A}, \gamma, D, \theta, \boldsymbol{\phi})$, consisting of finite state and action sets $\mathcal{S}$ and $\mathcal{A}$, discount factor $\gamma$, initial state distribution $D$, transition function $\theta(s, a, s') \triangleq \Pr(s_{t+1} = s' \mid s_t = s, a_t = a)$, and a set of $k$ features defined by the function $\boldsymbol{\phi} : \mathcal{S} \to \mathbb{R}^k$.

The true reward function $R^*$ is unknown. For ease of exposition, we assume that $R^*(s) = \mathbf{w}^* \cdot \boldsymbol{\phi}(s)$, for some $\mathbf{w}^* \in \mathbb{R}^k$, although we also show how our analysis extends to the case when this does not hold.

For any policy $\pi$ in $M$, the value of $\pi$ (with respect to the initial state distribution) is defined by

$$V(\pi) \triangleq E\left[\sum_{t=0}^{\infty} \gamma^t R^*(s_t) \,\Big|\, \pi, \theta, D\right].$$

where the initial state $s_0$ is chosen according to $D$, and the remaining states are chosen according to $\pi$ and $\theta$. We also define a $k$-length *feature expectations* vector,

$$\boldsymbol{\mu}(\pi) \triangleq E\left[\sum_{t=0}^{\infty} \gamma^t \boldsymbol{\phi}(s_t) \;\middle|\; \pi, \theta, D\right].$$

From its definition, it should be clear that "feature expectations" is a (somewhat misleading) abbreviation for "expected, cumulative, discounted feature values." Importantly, since $R^*(s) = \mathbf{w}^* \cdot \boldsymbol{\phi}(s)$, we have $V(\pi) = \mathbf{w}^* \cdot \boldsymbol{\mu}(\pi)$, by linearity of expectation.

We say that a feature expectations vector $\hat{\boldsymbol{\mu}}$ is an $\epsilon$-*good* estimate of $\boldsymbol{\mu}(\pi)$ if $\|\hat{\boldsymbol{\mu}} - \boldsymbol{\mu}(\pi)\|_\infty \le \epsilon$. Likewise, we say that a policy $\hat{\pi}$ is $\epsilon$-*optimal* for $M$ if $|V(\hat{\pi}) - V(\pi^*)| \le \epsilon$, where $\pi^*$ is an optimal policy for $M$, i.e. $\pi^* = \arg\max_\pi V(\pi)$.[1]

We also assume that there is a policy $\pi_E$, called the *expert's policy*, which we are able to observe executing in $M$. Following Abbeel and Ng [1], our goal is to find a policy $\pi$ such that $V(\pi) \ge V(\pi_E) - \epsilon$, even though the true reward function $R^*$ is unknown. We also have the additional goal of finding a policy when *no* observations from the expert's policy are available. In that case, we find a policy that is optimal in a certain conservative sense.

Like Abbeel and Ng [1], the policy we find will not necessarily be stationary, but will instead be a *mixed policy*. A mixed policy $\boldsymbol{\psi}$ is a distribution over $\Pi$, the set of all *deterministic* stationary policies in $M$. Because $\Pi$ is finite (though extremely large), we can fix a numbering of the policies in $\Pi$, which we denote $\pi^1, \ldots, \pi^{|\Pi|}$. This allows us to treat $\boldsymbol{\psi}$ as a vector, where $\boldsymbol{\psi}(i)$ is the probability assigned to $\pi^i$. A mixed policy $\boldsymbol{\psi}$ is executed by randomly selecting the policy $\pi^i \in \Pi$ at time 0 with probability $\boldsymbol{\psi}(i)$, and exclusively following $\pi^i$ thereafter. It should be noted that the definitions of value and feature expectations apply to mixed policies as well: $V(\boldsymbol{\psi}) = E_{i \sim \boldsymbol{\psi}}[V(\pi^i)]$ and $\boldsymbol{\mu}(\boldsymbol{\psi}) = E_{i \sim \boldsymbol{\psi}}[\boldsymbol{\mu}(\pi^i)]$. Also note that mixed policies do not have any advantage over stationary policies in terms of value: if $\pi^*$ is an optimal stationary policy for $M$, and $\boldsymbol{\psi}^*$ is an optimal mixed policy, then $V(\boldsymbol{\psi}^*) = V(\pi^*)$.

The observations from the expert's policy $\pi_E$ are in the form of $m$ independent trajectories in $M$, each for simplicity of the same length $H$. A trajectory is just the sequence of states visited by the expert: $(s_0^i, s_1^i, \ldots, s_H^i)$ for the $i$th trajectory. Let $\boldsymbol{\mu}_E = \boldsymbol{\mu}(\pi_E)$ be the expert's feature expectations. We compute an estimate $\hat{\boldsymbol{\mu}}_E$ of $\boldsymbol{\mu}_E$ by averaging the observed feature values from the trajectories:

$$\hat{\boldsymbol{\mu}}_E = \frac{1}{m} \sum_{i=0}^{m} \sum_{t=0}^{H} \gamma^t \boldsymbol{\phi}(s_t^i).$$

## 3 Review of the Projection Algorithm

We compare our approach to the "projection algorithm" of Abbeel and Ng [1], which finds a policy that is at least as good as the expert's policy with respect to the unknown reward function.[2]

Abbeel and Ng [1] assume that $\boldsymbol{\phi}(s) \in [0,1]^k$, and that $R^*(s) = \mathbf{w}^* \cdot \boldsymbol{\phi}(s)$ for some $\mathbf{w}^* \in \mathbb{B}^k$, where $\mathbb{B}^k = \{\mathbf{w} : \|\mathbf{w}\|_1 \le 1\}$. Given $m$ independent trajectories from the expert's policy, the projection algorithm runs for $T$ iterations. It returns a mixed policy $\overline{\boldsymbol{\psi}}$ such that $\|\boldsymbol{\mu}(\overline{\boldsymbol{\psi}}) - \boldsymbol{\mu}_E\|_2 \le \epsilon$ as long as $T$ and $m$ are sufficiently large. In other words, their algorithm seeks to "match" the expert's feature expectations. The value of $\overline{\boldsymbol{\psi}}$ will necessarily be close to that of the expert's policy, since

$$
\begin{aligned}
|V(\overline{\boldsymbol{\psi}}) - V(\pi_E)| &= |\mathbf{w}^* \cdot \boldsymbol{\mu}(\overline{\boldsymbol{\psi}}) - \mathbf{w}^* \cdot \boldsymbol{\mu}_E| \\
&\le \|\mathbf{w}^*\|_2 \|\boldsymbol{\mu}(\overline{\boldsymbol{\psi}}) - \boldsymbol{\mu}_E\|_2 \qquad (1) \\
&\le \epsilon
\end{aligned}
$$

where in Eq. (1) we used the Cauchy-Schwartz inequality and $\|\mathbf{w}^*\|_2 \le \|\mathbf{w}^*\|_1 \le 1$.

The following theorem is the main result in Abbeel and Ng [1]. However, some aspects of their analysis are not covered by this theorem, such as the complexity of each iteration of the projection algorithm, and the sensitivity of the algorithm to various approximations. These are discussed immediately below.

**Theorem 1 (Abbeel and Ng [1]).** *Given an MDP\R, and $m$ independent trajectories from an expert's policy $\pi_E$. Suppose we execute the projection algorithm for $T$ iterations. Let $\overline{\psi}$ be the mixed policy returned by the algorithm. Then in order for*

$$|V(\overline{\psi}) - V(\pi_E)| \leq \epsilon \tag{2}$$

*to hold with probability at least $1 - \delta$, it suffices that*

$$T \geq O\left(\frac{k}{(\epsilon(1-\gamma))^2}\ln\frac{k}{\epsilon(1-\gamma)}\right)$$

*and*

$$m \geq \frac{2k}{(\epsilon(1-\gamma))^2}\ln\frac{2k}{\delta}.$$

We omit the details of the algorithm due to space constraints, but note that each iteration involves only two steps that are computationally expensive:

1. Find an optimal policy with respect to a given reward function.
2. Compute the feature expectations of a given policy.

The algorithm we present in Section 5 performs these same expensive tasks in each iteration, but requires far fewer iterations — just $O(\ln k)$ rather than $O(k \ln k)$, a tremendous savings when the number of features $k$ is large. Also, the projection algorithm has a post-processing step that requires invoking a quadratic program (QP) solver. Comparatively, the post-processing step for our algorithm is trivial.

Abbeel and Ng [1] provide several refinements of the analysis in Theorem 1. In particular, suppose that each sample trajectory has length $H \geq (1/(1-\gamma))\ln(1/(\epsilon_H(1-\gamma)))$, and that an $\epsilon_P$-optimal policy is found in each iteration of the projection algorithm (see Step 1 above). Also let $\epsilon_R = \min_{\mathbf{w}\in\mathbb{B}^k}\max_s|R^*(s) - \mathbf{w}\cdot\boldsymbol{\phi}(s)|$ be the "representation error" of the features. Abbeel and Ng [1] comment at various points in their paper that $\epsilon_H$, $\epsilon_P$, and $O(\epsilon_R)$ should be added to the error bound of Theorem 1. In Section 5 we provide a unified analysis of these error terms in the context of our algorithm, and also incorporate an $\epsilon_F$ term that accounts for computing an $\epsilon_F$-good feature expectations estimate in Step 2 above. We prove that our algorithm is sensitive to these error terms in a similar way as the projection algorithm.

## 4 Apprenticeship Learning via Game Playing

Notice the two-sided bound in Theorem 1: the theorem guarantees that the apprentice will do almost as well as the expert, but *also almost as badly*. This is because the value of a policy is a linear combination of its feature expectations, and the goal of the projection algorithm is to match the expert's feature expectations.

We will take a different approach. We assume that $\boldsymbol{\phi}(s) \in [-1, 1]^k$, and that $R^*(s) = \mathbf{w}^* \cdot \boldsymbol{\phi}(s)$ for some $\mathbf{w}^* \in \mathbb{S}^k$, where $\mathbb{S}^k = \{\mathbf{w} \in \mathbb{R}^k : \|\mathbf{w}\|_1 = 1$ and $\mathbf{w} \succeq \mathbf{0}\}$.[3] The impact of this minor change in the domains of $\mathbf{w}$ and $\boldsymbol{\phi}$ is discussed further in Section 5.2. Let $\boldsymbol{\Psi}$ be the set of all mixed policies in $M$. Now consider the optimization

$$v^* = \max_{\boldsymbol{\psi}\in\boldsymbol{\Psi}} \min_{\mathbf{w}\in\mathbb{S}^k} \left[\mathbf{w}\cdot\boldsymbol{\mu}(\boldsymbol{\psi}) - \mathbf{w}\cdot\boldsymbol{\mu}_E\right]. \tag{3}$$

Our goal will be to find (actually, to approximate) the mixed policy $\psi^*$ that achieves $v^*$. Since $V(\psi) = \mathbf{w}^* \cdot \boldsymbol{\mu}(\psi)$ for all $\psi$, we have that $\psi^*$ is the policy in $\boldsymbol{\Psi}$ that maximizes $V(\psi) - V(\pi_E)$ with respect to the worst-case possibility for $\mathbf{w}^*$. Since $\mathbf{w}^*$ is unknown, maximizing for the worst-case is appropriate.

We begin by noting that, because $\mathbf{w}$ and $\boldsymbol{\psi}$ are both distributions, Eq. (3) is in the form of a two-person zero-sum *game*. Indeed, this is the motivation for redefining the domain of $\mathbf{w}$ as we did. The quantity $v^*$ is typically called the *game value*. In this game, the "min player" specifies a reward function by choosing $\mathbf{w}$, and the "max player" chooses a mixed policy $\boldsymbol{\psi}$. The goal of the min player is to cause the max player's policy to perform as poorly as possible relative to the expert, and the max player's goal is just the opposite. A game is defined by its associated *game matrix*. In our case, the game matrix is the $k \times |\Pi|$ matrix

$$\mathbf{G}(i,j) = \boldsymbol{\mu}^j(i) - \boldsymbol{\mu}_E(i) \tag{4}$$

where $\boldsymbol{\mu}(i)$ is the $i$th component of $\boldsymbol{\mu}$ and we have let $\boldsymbol{\mu}^j = \boldsymbol{\mu}(\pi^j)$ be the vector of feature expectations for the $j$th deterministic policy $\pi^j$. Now Eq. (3) can be rewritten in the form

$$v^* = \max_{\boldsymbol{\psi} \in \Psi} \min_{\mathbf{w} \in \mathbb{S}^k} \mathbf{w}^T \mathbf{G} \boldsymbol{\psi}. \tag{5}$$

In Eq. (3) and (5), the max player plays first, suggesting that the min player has an advantage. However, the well-known *minmax theorem* of von Neumann says that we can swap the min and max operators in Eq. (5) without affecting the game value. In other words,

$$v^* = \max_{\boldsymbol{\psi} \in \Psi} \min_{\mathbf{w} \in \mathbb{S}^k} \mathbf{w}^T \mathbf{G} \boldsymbol{\psi} = \min_{\mathbf{w} \in \mathbb{S}^k} \max_{\boldsymbol{\psi} \in \Psi} \mathbf{w}^T \mathbf{G} \boldsymbol{\psi}. \tag{6}$$

Finding $\boldsymbol{\psi}^*$ will not be useful unless we can establish that $v^* \geq 0$, i.e. that $\boldsymbol{\psi}^*$ will do at least as well as the expert's policy with respect to the worst-case possibility for $\mathbf{w}^*$. This fact is not immediately clear, since we are restricting ourselves to mixtures of deterministic policies, while we do not assume that the expert's policy is deterministic. However, note that in the rightmost expression in Eq. (6), the maximization over $\Psi$ is done after $\mathbf{w}$ — and hence the reward function — has been fixed. So the maximum is achieved by the best policy in $\Psi$ with respect to this fixed reward function. Note that if this is also an optimal policy, then $v^*$ will be nonnegative. It is well-known that in any MDP there always exists a deterministic optimal policy. Hence $v^* \geq 0$.

In fact, we may have $v^* > 0$. Suppose it happens that $\boldsymbol{\mu}(\boldsymbol{\psi}^*) \succ \boldsymbol{\mu}(\pi_E)$. Then $\boldsymbol{\psi}^*$ will dominate $\pi_E$, i.e. $\boldsymbol{\psi}^*$ will have higher value than $\pi_E$ regardless of the actual value of $\mathbf{w}^*$, because we assumed that $\mathbf{w}^* \succeq \mathbf{0}$. Essentially, by assuming that each component of the true weight vector is nonnegative, we are assuming that we have correctly specified the "sign" of each feature. This means that, other things being equal, a larger value for each feature implies a larger reward.

So when $v^* > 0$, the mixed policy $\boldsymbol{\psi}^*$ to some extent ignores the expert, and instead exploits prior knowledge about the true reward function encoded by the features. We present experimental results that explore this aspect of our approach in Section 7.

## 5 The Multiplicative Weights for Apprenticeship Learning (MWAL) Algorithm

In the previous section, we motivated the goal of finding the mixed policy $\boldsymbol{\psi}^*$ that achieves the maximum in Eq. (3) (or equivalently, in Eq. (5)). In this section we present an efficient algorithm for solving this optimization problem.

Recall the game formulated in the previous section. In the terminology of game theory, $\mathbf{w}$ and $\boldsymbol{\psi}$ are called *strategies* for the min and max player respectively , and $\boldsymbol{\psi}^*$ is called an optimal strategy for the max player. Also, a strategy $\dot{\mathbf{w}}$ is called *pure* if $\dot{\mathbf{w}}(i) = 1$ for some $i$.

Typically, one finds an optimal strategy for a two-player zero-sum game by solving a linear program. However, the complexity of that approach scales with the size of the game matrix. In our case, the game matrix $\mathbf{G}$ is huge, since it has as many columns as the number of deterministic policies in the MDP\R.

Freund and Schapire [2] described a multiplicative weights algorithm for finding approximately optimal strategies in games with large or even unknown game matrices. To apply their algorithm to a game matrix $\mathbf{G}$, it suffices to be able to efficiently perform the following two steps:

1. Given a min player strategy $\mathbf{w}$, find $\arg\max_{\boldsymbol{\psi} \in \Psi} \mathbf{w}^T \mathbf{G} \boldsymbol{\psi}$.

2. Given a max player strategy $\psi$, compute $\dot{\mathbf{w}}^T \mathbf{G} \psi$ for each pure strategy $\dot{\mathbf{w}}$.

Observe that these two steps are equivalent to the two steps of the projection algorithm from Section 3. Step 1 amounts to finding the optimal policy in a standard MDP with a known reward function. There are a huge array of techniques available for this, such as value iteration and policy iteration. Step 2 is the same as computing the feature expectations of a given policy. These can be computed exactly by solving $k$ systems of linear equations, or they can be approximated using iterative techniques. Importantly, the complexity of both steps scales with the size of the MDP\R, and not with the size of the game matrix $\mathbf{G}$.

Our Multiplicative Weights for Apprenticeship Learning (MWAL) algorithm is described below. Lines 7 and 8 of the algorithm correspond to Steps 1 and 2 directly above. The algorithm is essentially the MW algorithm of Freund and Schapire [2], applied to a game matrix very similar to $\mathbf{G}$.[4] We have also slightly extended their results to allow the MWAL algorithm, in lines 7 and 8, to estimate the optimal policy and its feature expectations, rather than requiring that they be computed exactly.

---

**Algorithm 1** The MWAL algorithm

1: **Given:** An MDP\R $M$ and an estimate of the expert's feature expectations $\hat{\boldsymbol{\mu}}_E$.
2: Let $\beta = \left(1 + \sqrt{\frac{2 \ln k}{T}}\right)^{-1}$.
3: Define $\widetilde{\mathbf{G}}(i, \boldsymbol{\mu}) \triangleq ((1 - \gamma)(\boldsymbol{\mu}(i) - \hat{\boldsymbol{\mu}}_E(i)) + 2)/4$, where $\boldsymbol{\mu} \in \mathbb{R}^k$.
4: Initialize $\mathbf{W}^{(1)}(i) = 1$ for $i = 1, \ldots, k$.
5: **for** $t = 1, \ldots, T$ **do**
6:     Set $\mathbf{w}^{(t)}(i) = \frac{\mathbf{W}^{(t)}(i)}{\sum_i \mathbf{W}^{(t)}(i)}$ for $i = 1, \ldots, k$.
7:     Compute an $\epsilon_P$-optimal policy $\hat{\pi}^{(t)}$ for $M$ with respect to reward function $R(s) = \mathbf{w}^{(t)} \cdot \boldsymbol{\phi}(s)$.
8:     Compute an $\epsilon_F$-good estimate $\hat{\boldsymbol{\mu}}^{(t)}$ of $\boldsymbol{\mu}^{(t)} = \boldsymbol{\mu}(\hat{\pi}^{(t)})$.
9:     $\mathbf{W}^{(t+1)}(i) = \mathbf{W}^{(t)}(i) \cdot \exp(\ln(\beta) \cdot \widetilde{\mathbf{G}}(i, \hat{\boldsymbol{\mu}}^{(t)}))$ for $i = 1, \ldots, k$.
10: **end for**
11: Post-processing: Return the mixed policy $\overline{\psi}$ that assigns probability $\frac{1}{T}$ to $\hat{\pi}^{(t)}$, for all $t \in \{1, \ldots, T\}$.

---

Theorem 2 below provides a performance guarantee for the mixed policy $\overline{\psi}$ returned by the MWAL algorithm, relative to the performance of the expert and the game value $v^*$. Its correctness is largely based on the main result in Freund and Schapire [2]. A proof is available in the supplement [4].

**Theorem 2.** *Given an MDP\R $M$, and $m$ independent trajectories from an expert's policy $\pi_E$. Suppose we execute the MWAL algorithm for $T$ iterations. Let $\overline{\psi}$ be the mixed policy returned by the algorithm. Let $\epsilon_F$ and $\epsilon_P$ be the approximation errors from lines 7 and 8 of the algorithm. Let $H \geq (1/(1 - \gamma)) \ln(1/(\epsilon_H(1 - \gamma)))$ be the length of each sample trajectory. Let $\epsilon_R = \min_{\mathbf{w} \in \mathbb{S}^k} \max_s |R^*(s) - \mathbf{w} \cdot \boldsymbol{\phi}(s)|$ be the representation error of the features. Let $v^* = \max_{\psi \in \Psi} \min_{\mathbf{w} \in \mathbb{S}^k} [\mathbf{w} \cdot \boldsymbol{\mu}(\psi) - \mathbf{w} \cdot \boldsymbol{\mu}_E]$ be the game value. Then in order for*

$$V(\overline{\psi}) \geq V(\pi_E) + v^* - \epsilon \tag{7}$$

*to hold with probability at least $1 - \delta$, it suffices that*

$$T \geq \frac{9 \ln k}{2(\epsilon'(1 - \gamma))^2} \tag{8}$$

$$m \geq \frac{2}{(\epsilon'(1 - \gamma))^2} \ln \frac{2k}{\delta} \tag{9}$$

$$\tag{10}$$

*where*

$$\epsilon' \leq \frac{\epsilon - (2\epsilon_F + \epsilon_P + 2\epsilon_H + 2\epsilon_R/(1 - \gamma))}{3}. \tag{11}$$

Note the differences between Theorem 1 and Theorem 2. Because $v^* \geq 0$, the guarantee of the MWAL algorithm in (7) is at least as strong as the guarantee of the projection algorithm in (2), and has the further benefit of being one-sided. Additionally, the iteration complexity of the MWAL algorithm is much lower. This not only implies a faster run time, but also implies that the mixed policy output by the MWAL algorithm consists of fewer stationary policies. And if a purely stationary policy is desired, it is not hard to show that the guarantee in (7) must hold for at least one of the stationary polices in the mixed policy (this is also true of the projection algorithm [1]).

The sample complexity in the Theorem 2 is also lower, but we believe that this portion of our analysis applies to the projection algorithm as well [Abbeel, personal communication], so the MWAL algorithm does not represent an improvement in this respect.

## 5.1 When no expert is available

Our game-playing approach can be very naturally and easily extended to the case where we do not have data from an expert. Instead of finding a policy that maximizes Eq. (3), we find a policy $\psi^*$ that maximizes

$$\max_{\psi \in \Psi} \min_{\mathbf{w} \in \mathbb{S}^k} \left[ \mathbf{w} \cdot \boldsymbol{\mu}(\psi) \right]. \tag{12}$$

Here $\psi^*$ is the best policy for the worst-case possibility for $\mathbf{w}^*$. The MWAL algorithm can be trivially adapted to find this policy just by setting $\boldsymbol{\mu}_E = \mathbf{0}$ (compare (12) to (3)).

The following corollary follows straightforwardly from the proof of Theorem 2.

**Corollary 1.** *Given an MDP\R $M$. Suppose we execute the 'no expert' version of the MWAL algorithm for $T$ iterations. Let $\overline{\psi}$ be the mixed policy returned by the algorithm. Let $\epsilon_F$, $\epsilon_P$, $\epsilon_R$ be defined as in Theorem 2. Let $v^* = \max_{\psi \in \Psi} \min_{\mathbf{w} \in \mathbb{S}^k} \left[ \mathbf{w} \cdot \boldsymbol{\mu}(\psi) \right]$. Then*

$$V(\overline{\psi}) \geq v^* - \epsilon \tag{13}$$

*if*

$$T \geq \frac{9 \ln k}{2(\epsilon'(1-\gamma))^2} \tag{14}$$

*where*

$$\epsilon' \leq \frac{\epsilon - (2\epsilon_F + \epsilon_P + 2\epsilon_R/(1-\gamma))}{3}. \tag{15}$$

## 5.2 Representation error

Although the MWAL algorithm makes different assumptions about the domains of $\mathbf{w}$ and $\boldsymbol{\phi}$ than the projection algorithm, these differences are of no real consequence. The same class of reward functions can be expressed under either set of assumptions by roughly doubling the number of features. Concretely, consider a feature function $\boldsymbol{\phi}$ that satisfies the assumptions of the projection algorithm. Then for each $s$, if $\boldsymbol{\phi}(s) = (f_1, \ldots, f_k)$, define $\boldsymbol{\phi}'(s) = (f_1, \ldots, f_k, -f_1, \ldots, -f_k, 0)$. Observe that $\boldsymbol{\phi}'$ satisfies the assumptions of the MWAL algorithm, and that $\min_{\mathbf{w} \in \mathbb{B}^k} \max_s |R^*(s) - \mathbf{w} \cdot \boldsymbol{\phi}(s)| \geq \min_{\mathbf{w} \in \mathbb{S}^{2k+1}} \max_s |R^*(s) - \mathbf{w} \cdot \boldsymbol{\phi}'(s)|$. So by only doubling the number of features, we can ensure that the representation error $\epsilon_R$ does not increase. Notably, employing this reduction forces the game value $v^*$ to be zero, ensuring that the MWAL algorithm, like the projection algorithm, will mimic the expert. This obsevation provides us with some useful guidance for selecting features for the MWAL algorithm: both the original and negated version of a feature should be used if we are uncertain how that feature is correlated with reward.

## 6 When the transition function is unknown

In the previous sections, as well as in Abbeel and Ng [1], it was assumed that the transition function $\theta(s, a, \cdot)$ was known. In this section we sketch how to remove this assumption. Our approach to applying the MWAL algorithm to this setting can be informally described as follows: Let $M = (\mathcal{S}, \mathcal{A}, \theta, \gamma, \phi)$ be the true MDP\R for which we are missing $\theta$. Consider the MLE estimate $\widehat{\theta}$ of $\theta$ that is formed from the expert's sample trajectories. Let $Z \subseteq \mathcal{S} \times \mathcal{A}$ be the set of state-action pairs that are visited "most frequently" by the expert. Then after observing enough trajectories, $\widehat{\theta}$ will

be an accurate estimate of $\theta$ on $Z$. We form a pessimistic estimate $\widehat{M}_Z$ of $M$ by using $\widehat{\theta}$ to model the transitions in $Z$, and route all other transitions to a special "dead state." Following Kearns and Singh [5], who used a very similar idea in their analyis of the $E^3$ algorithm, we call $\widehat{M}_Z$ the *induced MDP\R on Z*.

By a straightforward application of several technical lemmas due to Kearns and Singh [5] and Abbeel and Ng [6], it is possible to show that if the number of expert trajectories $m$ is at least $O(\frac{|\mathcal{S}|^3|\mathcal{A}|}{8\epsilon^3} \ln \frac{|\mathcal{S}|^3|\mathcal{A}|}{\delta\epsilon} + |\mathcal{S}||\mathcal{A}| \ln \frac{2|\mathcal{S}||\mathcal{A}|}{\delta})$, and we let $Z$ be the set of state-action pairs visited by the expert at least $O(\frac{|\mathcal{S}|^2}{4\epsilon^2} \ln \frac{|\mathcal{S}|^3|\mathcal{A}|}{\epsilon})$ times, then using $\widehat{M}_Z$ in place of $M$ in the MWAL algorithm will add only $O(\epsilon)$ to the error bound in Theorem 2. More details are available in the supplement [4], including a precise procedure for constructing $\widehat{M}_Z$.

# 7 Experiments

For ease of comparison, we tested the MWAL algorithm and the projection algorithm in a car driving simulator that resembled the experimental setup from Abbeel and Ng [1]. Videos of the experiments discussed below are available in the supplement [4].

In our simulator, the apprentice must navigate a car through randomly-generated traffic on a three-lane highway. We define three features for this environment: a collision feature (0 if contact with another car, and $1/2$ otherwise), an off-road feature (0 if on the grass, and $1/2$ otherwise), and a speed feature ($1/2$, $3/4$ and 1 for each of the three possible speeds, with higher values corresponding to higher speeds). Note that the features encode that, other things being equal, speed is good, and collisions and off-roads are bad.

|  | Fast Expert | Proj | MWAL | Bad Expert | Proj. | MWAL | No Expert | MWAL |
|---|---|---|---|---|---|---|---|---|
| Speed | Fast | Fast | Fast | Slow | Slow | Medium | - | Medium |
| Collisions (per sec) | 1.1 | 1.1 | 0.5 | 2.23 | 2.23 | 0 | - | 0 |
| Off-roads (per sec) | 0 | 0 | 0 | 8.0 | 8.0 | 0 | - | 0 |

The table above displays the results of using the MWAL and projection algorithms to learn a driving policy by observing two kinds of experts: a "fast" expert (drives at the fastest speed; indifferent to collisions), and a "bad" expert (drives at the slowest speed; tries to hit cars and go off-road). In both cases, the MWAL algorithm leverages information encoded in the features to produce a policy that is significantly better than the expert's policy.

We also applied the MWAL algorithm to the "no expert" setting (see Section 5.1). In that case, it produced a policy that drives as fast as possible without risking any collisions or off-roads. Given our features, this is indeed the best policy for the worst-case choice of reward function.

**Acknowledgments**

We thank Pieter Abbeel for his helpful explanatory comments regarding his proofs. We also thank the anonymous reviewers for their suggestions for additional experiments and other improvements. This work was supported by the NSF under grant IIS-0325500.

## Footnotes

[1] Note that this is weaker than the standard definition of optimality, as the policy only needs to be optimal with respect to the initial state distribution, and not necessarily at every state simultaneously.

[2] Abbeel and Ng [1] actually presented two algorithms for this task. Both had the same theoretical guarantees, but the projection algorithm is simpler and was empirically shown to be slightly faster.

[3]We use $\succeq$ to denote componentwise inequality. Likewise, we use $\succ$ to denote strict inequality in *every* component.

[4]Note that $\widetilde{\mathbf{G}}$ in Algorithm 1, in contrast to $\mathbf{G}$ in Eq. (4), depends on $\hat{\boldsymbol{\mu}}_E$ instead of $\boldsymbol{\mu}_E$. This is because $\boldsymbol{\mu}_E$ is unknown, and must be estimated. The other differences between $\widetilde{\mathbf{G}}$ and $\mathbf{G}$ are of no real consequence, and are further explained in the supplement [4].

# References

[1] P. Abbeel, A. Ng (2004). Apprenticeship Learning via Inverse Reinforcement Learning. *ICML* **21**.

[2] Y. Freund, R. E. Schapire (1999). Adaptive Game Playing Using Multiplicative Weights. *Games and Economic Behavior* **29**, 79–103.

[3] N. Ratliff, J. Bagnell, M. Zinkevich (2006). Maximum Margin Planning. *ICML* **23**.

[4] U. Syed, R. E. Schapire (2007). "A Game-Theoretic Approach to Apprenticeship Learning — Supplement". http://www.cs.princeton.edu/~usyed/nips2007/.

[5] M. Kearns, S. Singh (2002). Near-Optimal Reinforcement Learning in Polynomial Time. *Machine Learning* **49**, 209–232.

[6] P. Abbeel, A. Ng (2005). Exploration and Apprenticeship Learning in Reinforcement Learning. *ICML* **22**. (Long version; available at http://www.cs.stanford.edu/~pabbeel/)

